# Learning Bounds for Domain Adaptation

**John Blitzer, Koby Crammer, Alex Kulesza, Fernando Pereira, and Jennifer Wortman**
Department of Computer and Information Science
University of Pennsylvania, Philadelphia, PA 19146
{blitzer,crammer,kulesza,pereira,wortmanj}@cis.upenn.edu

## Abstract

Empirical risk minimization offers well-known learning guarantees when training and test data come from the same domain. In the real world, though, we often wish to adapt a classifier from a *source* domain with a large amount of training data to different *target* domain with very little training data. In this work we give uniform convergence bounds for algorithms that minimize a convex combination of source and target empirical risk. The bounds explicitly model the inherent trade-off between training on a large but inaccurate source data set and a small but accurate target training set. Our theory also gives results when we have multiple source domains, each of which may have a different number of instances, and we exhibit cases in which minimizing a non-uniform combination of source risks can achieve much lower target error than standard empirical risk minimization.

## 1   Introduction

Domain adaptation addresses a common situation that arises when applying machine learning to diverse data. We have ample data drawn from a *source* domain to train a model, but little or no training data from the *target* domain where we wish to use the model [17, 3, 10, 5, 9]. Domain adaptation questions arise in nearly every application of machine learning. In face recognition systems, training images are obtained under one set of lighting or occlusion conditions while the recognizer will be used under different conditions [14]. In speech recognition, acoustic models trained by one speaker need to be used by another [12]. In natural language processing, part-of-speech taggers, parsers, and document classifiers are trained on carefully annotated training sets, but applied to texts from different genres or styles [7, 6].

While many domain-adaptation algorithms have been proposed, there are only a few theoretical studies of the problem [3, 10]. Those studies focus on the case where training data is drawn from a source domain and test data is drawn from a different target domain. We generalize this approach to the case where we have some labeled data from the target domain in addition to a large amount of labeled source data. Our main result is a uniform convergence bound on the true target risk of a model trained to minimize a convex combination of empirical source and target risks. The bound describes an intuitive tradeoff between the quantity of the source data and the accuracy of the target data, and under relatively weak assumptions we can compute it from finite labeled and unlabeled samples of the source and target distributions. We use the task of sentiment classification to demonstrate that our bound makes correct predictions about model error with respect to a distance measure between source and target domains and the number of training instances.

Finally, we extend our theory to the case in which we have multiple sources of training data, each of which may be drawn according to a different distribution and may contain a different number of instances. Several authors have empirically studied a special case of this in which each *instance* is weighted separately in the loss function, and instance weights are set to approximate the target domain distribution [10, 5, 9, 11]. We give a uniform convergence bound for algorithms that min-

imize a convex combination of multiple empirical source risks and we show that these algorithms can outperform standard empirical risk minimization.

## 2 A Rigorous Model of Domain Adaptation

We formalize domain adaptation for binary classification as follows. A *domain* is a pair consisting of a distribution $\mathcal{D}$ on $\mathcal{X}$ and a labeling function $f : \mathcal{X} \to [0, 1]$.[1] Initially we consider two domains, a *source* domain $\langle \mathcal{D}_S, f_S \rangle$ and a *target* domain $\langle \mathcal{D}_T, f_T \rangle$.

A *hypothesis* is a function $h : \mathcal{X} \to \{0, 1\}$. The probability according the distribution $\mathcal{D}_S$ that a hypothesis $h$ disagrees with a labeling function $f$ (which can also be a hypothesis) is defined as

$$\epsilon_S(h, f) = \mathrm{E}_{\mathbf{x} \sim \mathcal{D}_S} [ |h(\mathbf{x}) - f(\mathbf{x})| ] .$$

When we want to refer to the *risk* of a hypothesis, we use the shorthand $\epsilon_S(h) = \epsilon_S(h, f_S)$. We write the empirical risk of a hypothesis on the source domain as $\hat{\epsilon}_S(h)$. We use the parallel notation $\epsilon_T(h, f)$, $\epsilon_T(h)$, and $\hat{\epsilon}_T(h)$ for the target domain.

We measure the distance between two distributions $\mathcal{D}$ and $\mathcal{D}'$ using a hypothesis class-specific distance measure. Let $\mathcal{H}$ be a hypothesis class for instance space $\mathcal{X}$, and $\mathcal{A}_{\mathcal{H}}$ be the set of subsets of $\mathcal{X}$ that are the support of some hypothesis in $\mathcal{H}$. In other words, for every hypothesis $h \in \mathcal{H}$, $\{x : x \in \mathcal{X}, h(x) = 1\} \in \mathcal{A}_{\mathcal{H}}$. We define the distance between two distributions as:

$$d_{\mathcal{H}}(\mathcal{D}, \mathcal{D}') = 2 \sup_{A \in \mathcal{A}_{\mathcal{H}}} |\mathrm{Pr}_{\mathcal{D}}[A] - \mathrm{Pr}_{\mathcal{D}'}[A]| .$$

For our purposes, the distance $d_{\mathcal{H}}$ has an important advantage over more common means for comparing distributions such as $L_1$ distance or the KL divergence: we can compute $d_{\mathcal{H}}$ from finite *unlabeled* samples of the distributions $\mathcal{D}$ and $\mathcal{D}'$ when $\mathcal{H}$ has finite VC dimension [4]. Furthermore, we can compute a finite-sample approximation to $d_{\mathcal{H}}$ by finding a classifier $h \in \mathcal{H}$ that maximally discriminates between (unlabeled) instances from $\mathcal{D}$ and $\mathcal{D}'$ [3].

For a hypothesis space $\mathcal{H}$, we define the symmetric difference hypothesis space $\mathcal{H} \Delta \mathcal{H}$ as

$$\mathcal{H} \Delta \mathcal{H} = \{h(\mathbf{x}) \oplus h'(\mathbf{x}) : h, h' \in \mathcal{H}\} ,$$

where $\oplus$ is the XOR operator. Each hypothesis $g \in \mathcal{H} \Delta \mathcal{H}$ labels as positive all points $x$ on which a given pair of hypotheses in $\mathcal{H}$ disagree. We can then define $\mathcal{A}_{\mathcal{H} \Delta \mathcal{H}}$ in the natural way as the set of all sets $A$ such that $A = \{x : x \in \mathcal{X}, h(x) \neq h'(x)\}$ for some $h, h' \in \mathcal{H}$. This allows us to define as above a distance $d_{\mathcal{H} \Delta \mathcal{H}}$ that satisfies the following useful inequality for any hypotheses $h, h' \in \mathcal{H}$, which is straight-forward to prove:

$$|\epsilon_S(h, h') - \epsilon_T(h, h')| \leq \frac{1}{2} d_{\mathcal{H} \Delta \mathcal{H}}(\mathcal{D}_S, \mathcal{D}_T) .$$

We formalize the difference between labeling functions by measuring error relative to other hypotheses in our class. The *ideal hypothesis* minimizes combined source and target risk:

$$h^* = \operatorname*{argmin}_{h \in \mathcal{H}} \epsilon_S(h) + \epsilon_T(h) .$$

We denote the combined risk of the ideal hypothesis by $\lambda = \epsilon_S(h^*) + \epsilon_T(h^*)$. The ideal hypothesis explicitly embodies our notion of adaptability. When the ideal hypothesis performs poorly, we cannot expect to learn a good target classifier by minimizing source error.[2] On the other hand, for the kinds of tasks mentioned in Section 1, we expect $\lambda$ to be small. If this is the case, we can reasonably approximate target risk using source risk and the distance between $\mathcal{D}_S$ and $\mathcal{D}_T$.

We illustrate the kind of result available in this setting with the following bound on the target risk in terms of the source risk, the difference between labeling functions $f_S$ and $f_T$, and the distance between the distributions $\mathcal{D}_S$ and $\mathcal{D}_T$. This bound is essentially a restatement of the main theorem of Ben-David et al. [3], with a small correction to the statement of their theorem.

**Theorem 1** *Let $\mathcal{H}$ be a hypothesis space of VC-dimension $d$ and $\mathcal{U}_S$, $\mathcal{U}_T$ be unlabeled samples of size $m'$ each, drawn from $\mathcal{D}_S$ and $\mathcal{D}_T$, respectively. Let $\hat{d}_{\mathcal{H}\Delta\mathcal{H}}$ be the empirical distance on $\mathcal{U}_S$, $\mathcal{U}_T$, induced by the symmetric difference hypothesis space. With probability at least $1 - \delta$ (over the choice of the samples), for every $h \in \mathcal{H}$,*

$$\epsilon_T(h) \le \epsilon_S(h) + \frac{1}{2}\hat{d}_{\mathcal{H}\Delta\mathcal{H}}(\mathcal{U}_S,\mathcal{U}_T) + 4\sqrt{\frac{2d\log(2m') + \log(\frac{4}{\delta})}{m'}} + \lambda \ .$$

The corrected proof of this result can be found Appendix A.[3] The main step in the proof is a variant of the triangle inequality in which the sides of the triangle represent errors between different decision rules [3, 8]. The bound is relative to $\lambda$. When the combined error of the ideal hypothesis is large, there is no classifier that performs well on both the source and target domains, so we cannot hope to find a good target hypothesis by training only on the source domain. On the other hand, for small $\lambda$ (the most relevant case for domain adaptation), Theorem 1 shows that source error and unlabeled $\mathcal{H}\Delta\mathcal{H}$-distance are important quantities for computing target error.

## 3  A Learning Bound Combining Source and Target Data

Theorem 1 shows how to relate source and target risk. We now proceed to give a learning bound for empirical risk minimization using combined source and target training data. In order to simplify the presentation of the trade-offs that arise in this scenario, we state the bound in terms of VC dimension. Similar, tighter bounds could be derived using more sophisticated measures of complexity such as PAC-Bayes [15] or Rademacher complexity [2] in an analogous way.

At train time a learner receives a sample $S = (S_T, S_S)$ of $m$ instances, where $S_T$ consists of $\beta m$ instances drawn independently from $\mathcal{D}_T$ and $S_S$ consists of $(1-\beta)m$ instances drawn independently from $\mathcal{D}_S$. The goal of a learner is to find a hypothesis that minimizes target risk $\epsilon_T(h)$. When $\beta$ is small, as in domain adaptation, minimizing empirical target risk may not be the best choice. We analyze learners that instead minimize a convex combination of empirical source and target risk:

$$\hat{\epsilon}_\alpha(h) = \alpha\hat{\epsilon}_T(h) + (1 - \alpha)\hat{\epsilon}_S(h)$$

We denote as $\epsilon_\alpha(h)$ the corresponding weighted combination of true source and target risks, measured with respect to $\mathcal{D}_S$ and $\mathcal{D}_T$.

We bound the target risk of a domain adaptation algorithm that minimizes $\hat{\epsilon}_\alpha(h)$. The proof of the bound has two main components, which we state as lemmas below. First we bound the difference between the target risk $\epsilon_T(h)$ and weighted risk $\epsilon_\alpha(h)$. Then we bound the difference between the true and empirical weighted risks $\epsilon_\alpha(h)$ and $\hat{\epsilon}_\alpha(h)$. The proofs of these lemmas, as well as the proof of Theorem 2, are in Appendix B.

**Lemma 1** *Let $h$ be a hypothesis in class $\mathcal{H}$. Then*

$$|\epsilon_\alpha(h) - \epsilon_T(h)| \le (1 - \alpha)\left(\frac{1}{2}d_{\mathcal{H}\Delta\mathcal{H}}(\mathcal{D}_S, \mathcal{D}_T) + \lambda\right) \ .$$

The lemma shows that as $\alpha$ approaches 1, we rely increasingly on the target data, and the distance between domains matters less and less. The proof uses a similar technique to that of Theorem 1.

**Lemma 2** *Let $\mathcal{H}$ be a hypothesis space of VC-dimension $d$. If a random labeled sample of size $m$ is generated by drawing $\beta m$ points from $\mathcal{D}_T$ and $(1 - \beta)m$ points from $\mathcal{D}_S$, and labeling them according to $f_S$ and $f_T$ respectively, then with probability at least $1 - \delta$ (over the choice of the samples), for every $h \in \mathcal{H}$*

$$|\hat{\epsilon}_\alpha(h) - \epsilon_\alpha(h)| < \sqrt{\frac{\alpha^2}{\beta} + \frac{(1 - \alpha)^2}{1 - \beta}}\sqrt{\frac{d\log(2m) - \log\delta}{2m}} \ .$$

The proof is similar to standard uniform convergence proofs [16, 1], but it uses Hoeffding's inequality in a different way because the bound on the range of the random variables underlying the inequality varies with $\alpha$ and $\beta$. The lemma shows that as $\alpha$ moves away from $\beta$ (where each instance is weighted equally), our finite sample approximation to $\epsilon_\alpha(h)$ becomes less reliable.

**Theorem 2** *Let $\mathcal{H}$ be a hypothesis space of VC-dimension $d$. Let $\mathcal{U}_S$ and $\mathcal{U}_T$ be unlabeled samples of size $m'$ each, drawn from $\mathcal{D}_S$ and $\mathcal{D}_T$ respectively. Let $S$ be a labeled sample of size $m$ generated by drawing $\beta m$ points from $\mathcal{D}_T$ and $(1-\beta)m$ points from $\mathcal{D}_S$, labeling them according to $f_S$ and $f_T$, respectively. If $\hat{h} \in \mathcal{H}$ is the empirical minimizer of $\hat{\epsilon}_\alpha(h)$ on $S$ and $h_T^* = \min_{h \in \mathcal{H}} \epsilon_T(h)$ is the target risk minimizer, then with probability at least $1-\delta$ (over the choice of the samples),*

$$\epsilon_T(\hat{h}) \leq \epsilon_T(h_T^*) + 2\sqrt{\frac{\alpha^2}{\beta} + \frac{(1-\alpha)^2}{1-\beta}}\sqrt{\frac{d\log(2m) - \log\delta}{2m}} +$$

$$2(1-\alpha)\left(\frac{1}{2}\hat{d}_{\mathcal{H}\Delta\mathcal{H}}(\mathcal{U}_S, \mathcal{U}_T) + 4\sqrt{\frac{2d\log(2m') + \log(\frac{4}{\delta})}{m'}} + \lambda\right) .$$

When $\alpha = 0$ (that is, we ignore target data), the bound is identical to that of Theorem 1, but with an empirical estimate for the source error. Similarly when $\alpha = 1$ (that is, we use only target data), the bound is the standard learning bound using only target data. At the optimal $\alpha$ (which minimizes the right hand side), the bound is always at least as tight as either of these two settings. Finally note that by choosing different values of $\alpha$, the bound allows us to effectively trade off the small amount of target data against the large amount of less relevant source data.

We remark that when it is known that $\lambda = 0$, the dependence on $m$ in Theorem 2 can be improved; this corresponds to the restricted or realizable setting.

## 4    Experimental Results

We evaluate our theory by comparing its predictions to empirical results. While ideally Theorem 2 could be directly compared with test error, this is not practical because $\lambda$ is unknown, $d_{\mathcal{H}\Delta\mathcal{H}}$ is computationally intractable [3], and the VC dimension $d$ is too large to be a useful measure of complexity. Instead, we develop a simple approximation of Theorem 2 that we can compute from unlabeled data. For many adaptation tasks, $\lambda$ is small (there exists a classifier which is simultaneously good for both domains), so we ignore it here. We approximate $d_{\mathcal{H}\Delta\mathcal{H}}$ by training a linear classifier to discriminate between the two domains. We use a standard hinge loss (normalized by dividing by the number of instances) and apply the quantity $1 - \big(\text{hinge loss}\big)$ in place of the actual $d_{\mathcal{H}\Delta\mathcal{H}}$. Let $\zeta(\mathcal{U}_S, \mathcal{U}_T)$ be our approximation to $d_{\mathcal{H}\Delta\mathcal{H}}$, computed from source and target unlabeled data. For domains that can be perfectly separated with margin, $\zeta(\mathcal{U}_S, \mathcal{U}_T) = 1$. For domains that are indistinguishable, $\zeta(\mathcal{U}_S, \mathcal{U}_T) = 0$. Finally we replace the VC dimension sample complexity term with a tighter constant $C$. The resulting approximation to the bound of Theorem 2 is

$$f(\alpha) = \sqrt{\frac{C}{m}\left(\frac{\alpha^2}{\beta} + \frac{(1-\alpha)^2}{1-\beta}\right)} + (1-\alpha)\zeta(\mathcal{U}_S, \mathcal{U}_T) . \tag{1}$$

Our experimental results are for the task of sentiment classification. Sentiment classification systems have recently gained popularity because of their potential applicability to a wide range of documents in many genres, from congressional records to financial news. Because of the large number of potential genres, sentiment classification is an ideal area for domain adaptation. We use the data provided by Blitzer et al. [6], which consists of reviews of eight types of products from Amazon.com: apparel, books, DVDs, electronics, kitchen appliances, music, video, and a catchall category "other". The task is binary classification: given a review, predict whether it is positive (4 or 5 out of 5 stars) or negative (1 or 2 stars). We chose the "apparel" domain as our target domain, and all of the plots on the right-hand side of Figure 1 are for this domain. We obtain empirical curves for the error as a function of $\alpha$ by training a classifier using a weighted hinge loss. Suppose the target domain has weight $\alpha$ and there are $\beta m$ target training instances. Then we scale the loss of target training instance by $\alpha/\beta$ and the loss of a source training instance by $(1-\alpha)/(1-\beta)$.

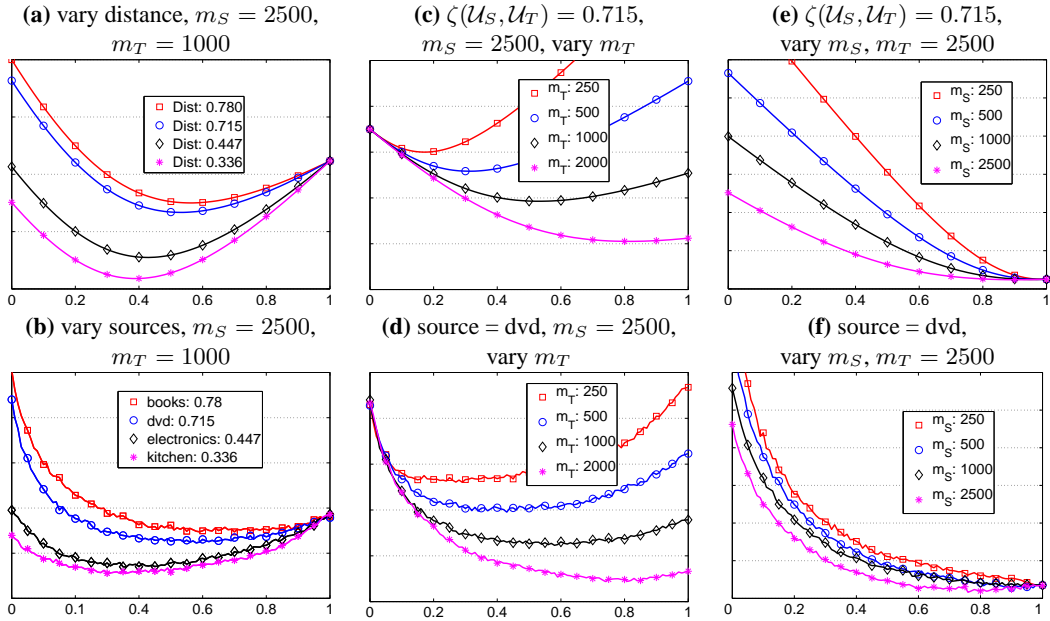

Figure 1: Comparing the bound with test error for sentiment classification. The $x$-axis of each figure shows $\alpha$. The $y$-axis shows the value of the bound or test set error. (a), (c), and (e) depict the bound, (b), (d), and (f) the test error. Each curve in (a) and (b) represents a different distance. Curves in (c) and (d) represent different numbers of target instances. Curves in (e) and (f) represent different numbers of source instances.

Figure 1 shows a series of plots of equation 1 (on the top) coupled with corresponding plots of test error (on the bottom) as a function of $\alpha$ for different amounts of source and target data and different distances between domains. In each pair of plots, a single parameter (distance, number of target instances $m_T$, or number of source instances $m_S$) is varied while the other two are held constant. Note that $\beta = m_T/(m_T + m_S)$. The plots on the top part of Figure 1 are not meant to be numerical proxies for the true error (For the source domains "books" and "dvd", the distance alone is well above $\frac{1}{2}$). Instead, they are scaled to illustrate that the bound is similar in shape to the true error curve and that relative relationships are preserved. By choosing a different $C$ in equation 1 for each curve, one can achieve complete control over their minima. In order to avoid this, we only use a single value of $C = 1600$ for all 12 curves on the top part of Figure 1.

First note that in every pair of plots, the empirical error curves have a roughly convex shape that mimics the shape of the bounds. Furthermore the value of $\alpha$ which minimizes the bound also has a low empirical error for each corresponding curve. This suggests that choosing $\alpha$ to minimize the bound of Theorem 2 and subsequently training a classifier to minimize the empirical error $\hat{\epsilon}_\alpha(h)$ can work well in practice, provided we have a reasonable measure of complexity.[4] Figures 1a and 1b show that more distant source domains result in higher target error. Figures 1c and 1d illustrate that for more target data, we have not only lower error in general, but also a higher minimizing $\alpha$. Finally, figures 1e and 1f depict the limitation of distant source data. With enough target data, no matter how much source data we include, we always prefer to use only the target data. This is reflected in our bound as a phase transition in the value of the optimal $\alpha$ (governing the tradeoff between source and target data). The phase transition occurs when $m_T = C/\zeta(\mathcal{U}_S, \mathcal{U}_T)^2$ (See Figure 2).

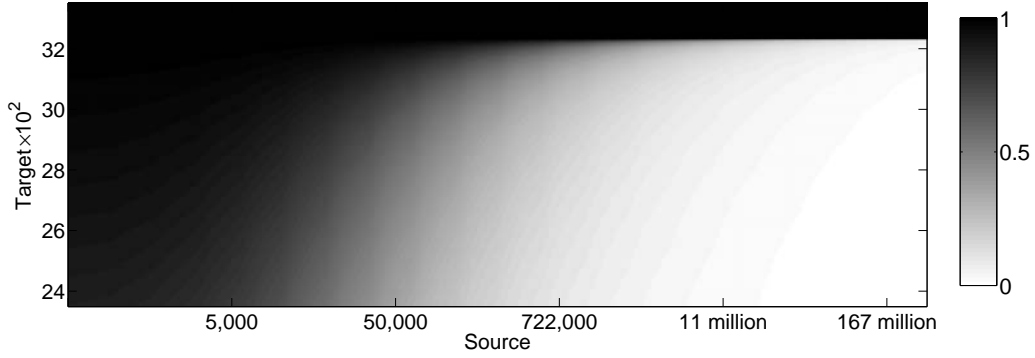

Figure 2: An example of the phase transition in the optimal $\alpha$. The value of $\alpha$ which minimizes the bound is indicated by the intensity, where black means $\alpha = 1$ (corresponding to ignoring source and learning only from target data). We fix $C = 1600$ and $\zeta(\mathcal{U}_S, \mathcal{U}_T) = 0.715$, as in our sentiment results. The $x$-axis shows the number of source instances (log-scale). The $y$-axis shows the number of target instances. A phase transition occurs at 3,130 target instances. With more target instances than this, it is more effective to ignore even an infinite amount of source data.

## 5 Learning from Multiple Sources

We now explore an extension of our theory to the case of multiple source domains. We are presented with data from $N$ distinct sources. Each source $S_j$ is associated with an unknown underlying distribution $\mathcal{D}_j$ over input points and an unknown labeling function $f_j$. From each source $S_j$, we are given $m_j$ labeled training instances, and our goal is to use these instances to train a model to perform well on a target domain $\langle \mathcal{D}_T, f_T \rangle$, which may or may not be one of the sources. This setting is motivated by several new domain adaptation algorithms [10, 5, 11, 9] that weigh the loss from training instances depending on how "far" they are from the target domain. That is, each training instance is its own source domain.

As in the previous sections, we will examine algorithms that minimize convex combinations of training errors over the labeled examples from each source domain. As before, we let $m_j = \beta_j m$ with $\sum_{j=1}^N \beta_j = 1$. Given a vector $\boldsymbol{\alpha} = (\alpha_1, \cdots, \alpha_N)$ of domain weights with $\sum_j \alpha_j = 1$, we define the empirical $\boldsymbol{\alpha}$-weighted error of function $h$ as

$$\hat{\epsilon}_{\boldsymbol{\alpha}}(h) = \sum_{j=1}^N \alpha_j \hat{\epsilon}_j(h) = \sum_{j=1}^N \frac{\alpha_j}{m_j} \sum_{x \in S_j} |h(x) - f_j(x)| \ .$$

The true $\boldsymbol{\alpha}$-weighted error $\epsilon_{\boldsymbol{\alpha}}(h)$ is defined analogously. Let $\mathcal{D}_{\boldsymbol{\alpha}}$ be a mixture of the $N$ source distributions with mixing weights equal to the components of $\boldsymbol{\alpha}$. Finally, analogous to $\lambda$ in the single-source setting, we define the error of the multi-source ideal hypothesis for a weighting $\boldsymbol{\alpha}$ as

$$\gamma_{\boldsymbol{\alpha}} = \min_h \{\epsilon_T(h) + \epsilon_{\boldsymbol{\alpha}}(h)\} = \min_h \{\epsilon_T(h) + \sum_{j=1}^N \alpha_j \epsilon_j(h)\} \ .$$

The following theorem gives a learning bound for empirical risk minimization using the empirical $\boldsymbol{\alpha}$-weighted error.

**Theorem 3** *Suppose we are given $m_j$ labeled instances from source $S_j$ for $j = 1 \ldots N$. For a fixed vector of weights $\boldsymbol{\alpha}$, let $\hat{h} = \operatorname{argmin}_{h \in \mathcal{H}} \hat{\epsilon}_{\boldsymbol{\alpha}}(h)$, and let $h_T^* = \operatorname{argmin}_{h \in \mathcal{H}} \epsilon_T(h)$. Then for any $\delta \in (0, 1)$, with probability at least $1 - \delta$ (over the choice of samples from each source),*

$$\epsilon_T(\hat{h}) \leq \epsilon_T(h_T^*) + 2\sqrt{\sum_{j=1}^N \frac{\alpha_j^2}{\beta_j}} \sqrt{\frac{d \log 2m - \log \delta}{2m}} + 2\left(\gamma_{\boldsymbol{\alpha}} + \frac{1}{2} d_{\mathcal{H}\Delta\mathcal{H}}(D_{\boldsymbol{\alpha}}, D_T)\right) \ .$$

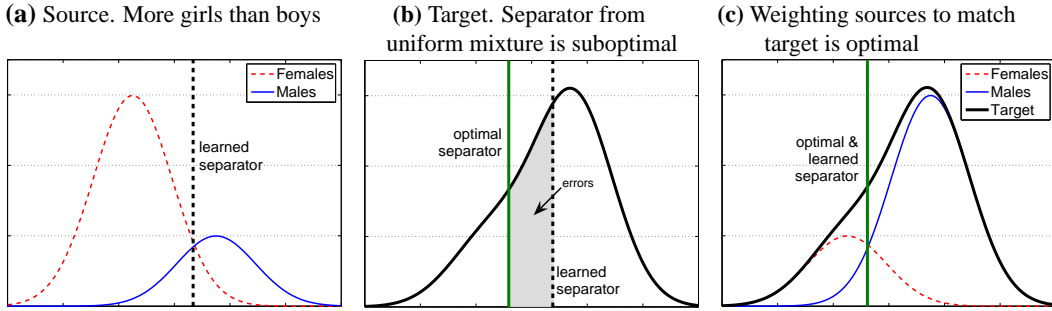

**(a)** Source. More girls than boys     **(b)** Target. Separator from     **(c)** Weighting sources to match
                               uniform mixture is suboptimal            target is optimal

Figure 3: A 1-dimensional example illustrating how non-uniform mixture weighting can result in optimal error. We observe one feature, which we use to predict gender. **(a)** At train time we observe more females than males. **(b)** Learning by uniformly weighting the training data causes us to learn a suboptimal decision boundary, **(c)** but by weighting the males more highly, we can match the target data and learn an optimal classifier.

The full proof is in appendix C. Like the proof of Theorem 2, it is split into two parts. The first part bounds the difference between the $\boldsymbol{\alpha}$-weighted error and the target error similar to lemma 1. The second is a uniform convergence bound for $\hat{\epsilon}_{\boldsymbol{\alpha}}(h)$ similar to lemma 2.

Theorem 3 reduces to Theorem 2 when we have only two sources, one of which is the target domain (that is, we have some small number of target instances). It is more general, though, because by manipulating $\boldsymbol{\alpha}$ we can effectively change the source domain. This has two consequences. First, we demand that there exists a hypothesis $h^*$ which has low error on both the $\boldsymbol{\alpha}$-weighted convex combination of sources and the target domain. Second, we measure distance between the target and a mixture of sources, rather than between the target and a single source.

One question we might ask is whether there exist settings where a non-uniform weighting can lead to a significantly lower value of the bound than a uniform weighting. This can happen if some non-uniform weighting of sources accurately approximates the target domain. As a hypothetical example, suppose we are trying to predict gender from height (Figure 3). Each instance is drawn from a gender-specific Gaussian. In this example, we can find the optimal classifier by weighting the "males" and "females" components of the source to match the target.

## 6  Related Work

Domain adaptation is a widely-studied area, and we cannot hope to cover every aspect and application of it here[5]. Instead, in this section we focus on other theoretical approaches to domain adaptation. While we do not explicitly address the relationship in this paper, we note that domain adaptation is closely related to the setting of covariate shift, which has been studied in statistics. In addition to the work of Huang et al. [10], several other authors have considered learning by assigning separate weights to the components of the loss function corresponding to separate instances. Bickel at al. [5] and Jiang and Zhai [11] suggest promising empirical algorithms that in part inspire our Theorem 3. We hope that our work can help to explain when these algorithms are effective. Dai et al. [9] considered weighting instances using a transfer-aware variant of boosting, but the learning bounds they give are no stronger than bounds which completely ignore the source data.

Crammer et al. [8] consider learning when the marginal distribution on instances is the same across sources but the labeling function may change. This corresponds in our theory to cases where $d_{\mathcal{H}\Delta\mathcal{H}} = 0$ but $\lambda$ is large. Like us they consider multiple sources, but their notion of weighting is less general. They consider only including or discarding a source entirely.

Li and Bilmes [13] give PAC-Bayesian learning bounds for adaptation using "divergence priors". They place source-centered prior on the parameters of a model learned in the target domain. Like

our model, the divergence prior also emphasizes the tradeoff between source and target. In our model, though, we measure the divergence (and consequently the bias) of the source domain from unlabeled data. This allows us to choose the best tradeoff between source and target labeled data.

## 7 Conclusion

In this work we investigate the task of domain adaptation when we have a large amount of training data from a source domain but wish to apply a model in a target domain with a much smaller amount of training data. Our main result is a uniform convergence learning bound for algorithms which minimize convex combinations of source and target empirical risk. Our bound reflects the trade-off between the size of the source data and the accuracy of the target data, and we give a simple approximation to it that is computable from finite labeled and unlabeled samples. This approximation makes correct predictions about model test error for a sentiment classification task. Our theory also extends in a straightforward manner to a multi-source setting, which we believe helps to explain the success of recent empirical work in domain adaptation.

Our future work has two related directions. First, we wish to tighten our bounds, both by considering more sophisticated measures of complexity [15, 2] and by focusing our distance measure on the most relevant features, rather than all the features. We also plan to investigate algorithms that choose a convex combination of multiple sources to minimize the bound in Theorem 3.

## 8 Acknowledgements

This material is based upon work partially supported by the Defense Advanced Research Projects Agency (DARPA) under Contract No. NBCHD030010. Any opinions, findings, and conclusions or recommendations expressed in this material are those of the authors and do not necessarily reflect the views of the DARPA or Department of Interior-National Business Center (DOI-NBC).

## Footnotes

[1]This notion of domain is not the domain of a function. To avoid confusion, we will always mean a specific distribution and function pair when we say domain.

[2]Of course it is still possible that the source data contains relevant information about the target function even when the ideal hypothesis performs poorly — suppose, for example, that $f_S(x) = 1$ if and only if $f_T(x) = 0$ — but a classifier trained using source data will perform poorly on data from the target domain in this case.

[3]A longer version of this paper that includes the omitted appendix can be found on the authors' websites.

[4]Although Theorem 2 does not hold uniformly for all $\alpha$ as stated, this is easily remedied via an application of the union bound. The resulting bound will contain an additional logarithmic factor in the complexity term.

[5]The NIPS 2006 Workshop on Learning When Test and Training Inputs have Different Distributions (http://ida.first.fraunhofer.de/projects/different06/) contains a good set of references on domain adaptation and related topics.

## References

[1] M. Anthony and P. Bartlett. *Neural Network Learning: Theoretical Foundations*. Cambridge University Press, Cambridge, 1999.

[2] P. Barlett and S. Mendelson. Rademacher and gaussian complexities: Risk bounds and structural results. *JMLR*, 3:463–482, 2002.

[3] S. Ben-David, J. Blitzer, K. Crammer, and F. Pereira. Analysis of representations for domain adaptation. In *NIPS*, 2007.

[4] S. Ben-David, J. Gehrke, and D. Kifer. Detecting change in data streams. In *VLDB*, 2004.

[5] S. Bickel, M. Brückner, and T. Scheffer. Discriminative learning for differing training and test distributions. In *ICML*, 2007.

[6] J. Blitzer, M. Dredze, and F. Pereira. Biographies, bollywood, boomboxes and blenders: Domain adaptation for sentiment classification. In *ACL*, 2007.

[7] C. Chelba and A. Acero. Empirical methods in natural language processing. In *EMNLP*, 2004.

[8] K. Crammer, M. Kearns, and J. Wortman. Learning from multiple sources. In *NIPS*, 2007.

[9] W. Dai, Q. Yang, G. Xue, and Y. Yu. Boosting for transfer learning. In *ICML*, 2007.

[10] J. Huang, A. Smola, A. Gretton, K. Borgwardt, and B. Schoelkopf. Correcting sample selection bias by unlabeled data. In *NIPS*, 2007.

[11] J. Jiang and C. Zhai. Instance weighting for domain adaptation. In *ACL*, 2007.

[12] C. Legetter and P. Woodland. Maximum likelihood linear regression for speaker adaptation of continuous density hidden markov models. *Computer Speech and Language*, 9:171–185, 1995.

[13] X. Li and J. Bilmes. A bayesian divergence prior for classification adaptation. In *AISTATS*, 2007.

[14] A. Martinez. Recognition of partially occluded and/or imprecisely localized faces using a probabilistic approach. In *CVPR*, 2007.

[15] D. McAllester. Simplified PAC-Bayesian margin bounds. In *COLT*, 2003.

[16] V. Vapnik. *Statistical Learning Theory*. John Wiley, New York, 1998.

[17] P. Wu and T. Dietterich. Improving svm accuracy by training on auxiliary data sources. In *ICML*, 2004.

